# Conditional Visual Tracking in Kernel Space

**Cristian Sminchisescu**[1,2,3]    **Atul Kanujia**[3]    **Zhiguo Li**[3]    **Dimitris Metaxas**[3]

[1]TTI-C, 1497 East 50th Street, Chicago, IL, 60637, USA
[2]University of Toronto, Department of Computer Science, Canada
[3]Rutgers University, Department of Computer Science, USA
crismin@cs.toronto.edu, {kanaujia,zhli,dnm}@cs.rutgers.edu

## Abstract

We present a conditional temporal probabilistic framework for reconstructing 3D human motion in monocular video based on descriptors encoding image silhouette observations. For computational efficiency we restrict visual inference to low-dimensional kernel induced non-linear state spaces. Our methodology (kBME) combines kernel PCA-based non-linear dimensionality reduction (kPCA) and Conditional Bayesian Mixture of Experts (BME) in order to learn complex multivalued predictors between observations and model hidden states. This is necessary for accurate, inverse, visual perception inferences, where several probable, distant 3D solutions exist due to noise or the uncertainty of monocular perspective projection. Low-dimensional models are appropriate because many visual processes exhibit strong non-linear correlations in both the image observations and the target, hidden state variables. The learned predictors are temporally combined within a conditional graphical model in order to allow a principled propagation of uncertainty. We study several predictors and empirically show that the proposed algorithm positively compares with techniques based on regression, Kernel Dependency Estimation (KDE) or PCA alone, and gives results competitive to those of high-dimensional mixture predictors at a fraction of their computational cost. We show that the method successfully reconstructs the complex 3D motion of humans in real monocular video sequences.

## 1   Introduction and Related Work

We consider the problem of inferring 3D articulated human motion from monocular video. This research topic has applications for scene understanding including human-computer interfaces, markerless human motion capture, entertainment and surveillance. A monocular approach is relevant because in real-world settings the human body parts are rarely completely observed even when using multiple cameras. This is due to occlusions form other people or objects in the scene. A robust system has to necessarily deal with incomplete, ambiguous and uncertain measurements. Methods for 3D human motion reconstruction can be classified as *generative* and *discriminative*. They both require a state representation, namely a 3D human model with kinematics (joint angles) or shape (surfaces or joint positions) and they both use a set of image features as observations for state inference. The computational goal in both cases is the conditional distribution for the model state given

image observations.

Generative model-based approaches [6, 16, 14, 13] have been demonstrated to flexibly reconstruct complex unknown human motions and to naturally handle problem constraints. However it is difficult to construct reliable observation likelihoods due to the complexity of modeling human appearance. This varies widely due to different clothing and deformation, body proportions or lighting conditions. Besides being somewhat indirect, the generative approach further imposes strict conditional independence assumptions on the temporal observations given the states in order to ensure computational tractability. Due to these factors inference is expensive and produces highly multimodal state distributions [6, 16, 13]. Generative inference algorithms require complex annealing schedules [6, 13] or systematic non-linear search for local optima [16] in order to ensure continuing tracking.

These difficulties motivate the advent of a complementary class of discriminative algorithms [10, 12, 18, 2], that approximate the state conditional directly, in order to simplify inference. However, inverse, observation-to-state multivalued mappings are difficult to learn (see *e.g.* fig. 1a) and a probabilistic temporal setting is necessary. In an earlier paper [15] we introduced a probabilistic discriminative framework for human motion reconstruction. Because the method operates in the originally selected state and observation spaces that can be task generic, therefore redundant and often high-dimensional, inference is more expensive and can be less robust. To summarize, reconstructing 3D human motion in a

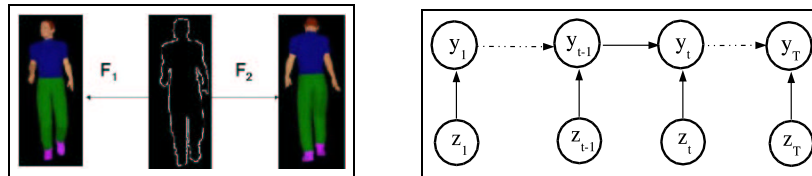

Figure 1: *(a, Left)* Example of $180^o$ ambiguity in predicting 3D human poses from silhouette image features (center). It is essential that multiple plausible solutions (*e.g.* $F_1$ and $F_2$) are correctly represented and tracked over time. A single state predictor will either average the distant solutions or zig-zag between them, see also tables 1 and 2. *(b, Right)* A conditional chain model. The local distributions $p(\mathbf{y}_t|\mathbf{y}_{t-1}, \mathbf{z}_t)$ or $p(\mathbf{y}_t|\mathbf{z}_t)$ are learned as in fig. 2. For inference, the predicted local state conditional is recursively combined with the filtered prior *c.f.* (1).

conditional temporal framework poses the following difficulties: (*i*) The mapping between temporal observations and states is multivalued (*i.e.* the local conditional distributions to be learned are multimodal), therefore it cannot be accurately represented using global function approximations. (*ii*) Human models have multivariate, high-dimensional continuous states of 50 or more human joint angles. The temporal state conditionals are multimodal which makes efficient Kalman filtering algorithms inapplicable. General inference methods (particle filters, mixtures) have to be used instead, but these are expensive for high-dimensional models (*e.g.* when reconstructing the motion of several people that operate in a joint state space). (*iii*) The components of the human state and of the silhouette observation vector exhibit strong correlations, because many repetitive human activities like walking or running have low intrinsic dimensionality. It appears wasteful to work with high-dimensional states of 50+ joint angles. Even if the space were truly high-dimensional, predicting correlated state dimensions independently may still be suboptimal.

In this paper we present a conditional temporal estimation algorithm that restricts visual inference to low-dimensional, kernel induced state spaces. To exploit correlations among observations and among state variables, we model the local, temporal conditional distributions using ideas from Kernel PCA [11, 19] and conditional mixture modeling [7, 5], here adapted to produce multiple probabilistic predictions. The corresponding predictor is

referred to as a *Conditional Bayesian Mixture of Low-dimensional Kernel-Induced Experts (kBME)*. By integrating it within a conditional graphical model framework (fig. 1b), we can exploit temporal constraints probabilistically. We demonstrate that this methodology is effective for reconstructing the 3D motion of multiple people in monocular video. Our contribution w.r.t. [15] is a probabilistic conditional inference framework that operates over a non-linear, kernel-induced low-dimensional state spaces, and a set of experiments (on both real and artificial image sequences) that show how the proposed framework positively compares with powerful predictors based on KDE, PCA, or with the high-dimensional models of [15] at a fraction of their cost.

## 2    Probabilistic Inference in a Kernel Induced State Space

We work with conditional graphical models with a chain structure [9], as shown in fig. 1b, These have continuous temporal states $\mathbf{y}_t$, $t = 1 \ldots T$, observations $\mathbf{z}_t$. For compactness, we denote joint states $\mathbf{Y}_t = (\mathbf{y}_1, \mathbf{y}_2, \ldots, \mathbf{y}_t)$ or joint observations $\mathbf{Z}_t = (\mathbf{z}_1, \ldots, \mathbf{z}_t)$. Learning and inference are based on local conditionals: $p(\mathbf{y}_t|\mathbf{z}_t)$ and $p(\mathbf{y}_t|\mathbf{y}_{t-1}, \mathbf{z}_t)$, with $\mathbf{y}_t$ and $\mathbf{z}_t$ being low-dimensional, kernel induced representations of some initial model having state $\mathbf{x}_t$ and observation $\mathbf{r}_t$. We obtain $\mathbf{z}_t, \mathbf{y}_t$ from $\mathbf{r}_t, \mathbf{x}_t$ using kernel PCA [11, 19]. *Inference is performed in a low-dimensional, non-linear, kernel induced latent state space* (see fig. 1b and fig. 2 and (1)). For display or error reporting, we compute the original conditional $p(\mathbf{x}|\mathbf{r})$, or a temporally filtered version $p(\mathbf{x}_t|\mathbf{R}_t)$, $\mathbf{R}_t = (\mathbf{r}_1, \mathbf{r}_2, \ldots, \mathbf{r}_t)$, using a learned pre-image state map [3].

### 2.1    Density Propagation for Continuous Conditional Chains

For online filtering, we compute the optimal distribution $p(\mathbf{y}_t|\mathbf{Z}_t)$ for the state $\mathbf{y}_t$, conditioned by observations $\mathbf{Z}_t$ up to time $t$. The filtered density can be recursively derived as:

$$p(\mathbf{y}_t|\mathbf{Z}_t) = \int_{\mathbf{y}_{t-1}} p(\mathbf{y}_t|\mathbf{y}_{t-1}, \mathbf{z}_t) p(\mathbf{y}_{t-1}|\mathbf{Z}_{t-1}) \tag{1}$$

We compute using a conditional mixture for $p(\mathbf{y}_t|\mathbf{y}_{t-1}, \mathbf{z}_t)$ (a Bayesian mixture of experts *c.f.* §2.2) and the prior $p(\mathbf{y}_{t-1}|\mathbf{Z}_{t-1})$, each having, say $M$ components. We integrate $M^2$ pairwise products of Gaussians analytically. The means of the expanded posterior are clustered and the centers are used to initialize a reduced $M$-component Kullback-Leibler approximation that is refined using gradient descent [15]. The propagation rule (1) is similar to the one used for discrete state labels [9], but here we work with multivariate continuous state spaces and represent the local multimodal state conditionals using kBME (fig. 2), and not log-linear models [9] (these would require intractable normalization). This complex continuous model rules out inference based on Kalman filtering or dynamic programming [9].

### 2.2    Learning Bayesian Mixtures over Kernel Induced State Spaces (kBME)

In order to model conditional mappings between low-dimensional non-linear spaces we rely on kernel dimensionality reduction and conditional mixture predictors. The authors of KDE [19] propose a powerful structured unimodal predictor. This works by decorrelating the output using kernel PCA and learning a ridge regressor between the input and each decorrelated output dimension.

Our procedure is also based on kernel PCA but takes into account the structure of the studied visual problem where both inputs and outputs are likely to be low-dimensional and the mapping between them multivalued. The output variables $\mathbf{x}_i$ are projected onto the column vectors of the principal space in order to obtain their principal coordinates $\mathbf{y}_i$. A

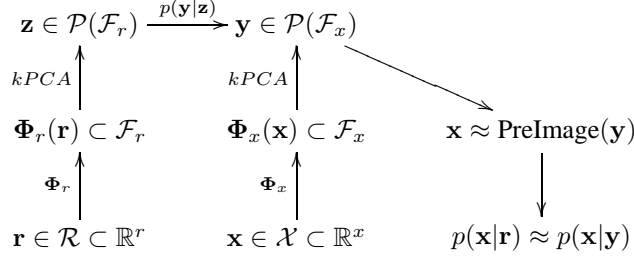

Figure 2: The learned low-dimensional predictor, kBME, for computing $p(\mathbf{x}|\mathbf{r}) \equiv p(\mathbf{x}_t|\mathbf{r}_t), \forall t$. (We similarly learn $p(\mathbf{x}_t|\mathbf{x}_{t-1}, \mathbf{r}_t)$, with input $(\mathbf{x}, \mathbf{r})$ instead of $\mathbf{r}$ – here we illustrate only $p(\mathbf{x}|\mathbf{r})$ for clarity.) The input $\mathbf{r}$ and the output $\mathbf{x}$ are decorrelated using Kernel PCA to obtain $\mathbf{z}$ and $\mathbf{y}$ respectively. The kernels used for the input and output are $\mathbf{\Phi}_r$ and $\mathbf{\Phi}_x$, with induced feature spaces $\mathcal{F}_r$ and $\mathcal{F}_x$, respectively. Their principal subspaces obtained by kernel PCA are denoted by $\mathcal{P}(\mathcal{F}_r)$ and $\mathcal{P}(\mathcal{F}_x)$, respectively. A conditional Bayesian mixture of experts $p(\mathbf{y}|\mathbf{z})$ is learned using the low-dimensional representation $(\mathbf{z}, \mathbf{y})$. Using learned local conditionals of the form $p(\mathbf{y}_t|\mathbf{z}_t)$ or $p(\mathbf{y}_t|\mathbf{y}_{t-1}, \mathbf{z}_t)$, temporal inference can be efficiently performed in a *low-dimensional kernel induced state space* (see *e.g.* (1) and fig. 1b). For visualization and error measurement, the filtered density, *e.g.* $p(\mathbf{y}_t|\mathbf{Z}_t)$, can be mapped back to $p(\mathbf{x}_t|\mathbf{R}_t)$ using the pre-image *c.f.* (3).

similar procedure is performed on the inputs $\mathbf{r}_i$ to obtain $\mathbf{z}_i$. In order to relate the reduced feature spaces of $\mathbf{z}$ and $\mathbf{y}$ ($\mathcal{P}(\mathcal{F}_r)$ and $\mathcal{P}(\mathcal{F}_x)$), we estimate a probability distribution over mappings from training pairs $(\mathbf{z}_i, \mathbf{y}_i)$. We use a conditional Bayesian mixture of experts (BME) [7, 5] in order to account for ambiguity when mapping similar, possibly identical reduced feature inputs to very different feature outputs, as common in our problem (fig. 1a). This gives a model that is a conditional mixture of low-dimensional kernel-induced experts (kBME):

$$p(\mathbf{y}|\mathbf{z}) = \sum_{j=1}^{M} g(\mathbf{z}|\boldsymbol{\delta}_j)\mathcal{N}(\mathbf{y}|\mathbf{W}_j\mathbf{z}, \boldsymbol{\Sigma}_j) \tag{2}$$

where $g(\mathbf{z}|\boldsymbol{\delta}_j)$ is a softmax function parameterized by $\boldsymbol{\delta}_j$ and $(\mathbf{W}_j, \boldsymbol{\Sigma}_j)$ are the parameters and the output covariance of expert $j$, here a linear regressor. As in many Bayesian settings [17, 5], the weights of the experts and of the gates, $\mathbf{W}_j$ and $\boldsymbol{\delta}_j$, are controlled by hierarchical priors, typically Gaussians with 0 mean, and having inverse variance hyperparameters controlled by a second level of Gamma distributions. We learn this model using a double-loop EM and employ ML-II type approximations [8, 17] with greedy (weight) subset selection [17, 15].

Finally, the kBME algorithm requires the computation of pre-images in order to recover the state distribution $\mathbf{x}$ from it's image $\mathbf{y} \in \mathcal{P}(\mathcal{F}_x)$. This is a closed form computation for polynomial kernels of odd degree. For more general kernels optimization or learning (regression based) methods are necessary [3]. Following [3, 19], we use a sparse Bayesian kernel regressor to learn the pre-image. This is based on training data $(\mathbf{x}_i, \mathbf{y}_i)$:

$$p(\mathbf{x}|\mathbf{y}) = \mathcal{N}(\mathbf{x}|\mathbf{A}\mathbf{\Phi}_y(\mathbf{y}), \boldsymbol{\Omega}) \tag{3}$$

with parameters and covariances $(\mathbf{A}, \boldsymbol{\Omega})$. Since temporal inference is performed in the low-dimensional kernel induced state space, the pre-image function needs to be calculated only for visualizing results or for the purpose of error reporting. Propagating the result from the reduced feature space $\mathcal{P}(\mathcal{F}_x)$ to the output space $\mathcal{X}$ pro-

duces a Gaussian mixture with $M$ elements, having coefficients $g(\mathbf{z}|\boldsymbol{\delta}_j)$ and components $\mathcal{N}(\mathbf{x}|\mathbf{A}\boldsymbol{\Phi}_y(\mathbf{W}_j\mathbf{z}), \mathbf{A}\mathbf{J}_{\boldsymbol{\Phi}_y}\boldsymbol{\Sigma}_j\mathbf{J}_{\boldsymbol{\Phi}_y}^\top\mathbf{A}^\top + \boldsymbol{\Omega})$, where $\mathbf{J}_{\boldsymbol{\Phi}_y}$ is the Jacobian of the mapping $\boldsymbol{\Phi}_y$.

## 3 Experiments

We run experiments on both real image sequences (fig. 5 and fig. 6) and on sequences where silhouettes were artificially rendered. The prediction error is reported in degrees (for mixture of experts, this is w.r.t. the most probable one, but see also fig. 4a), and normalized per joint angle, per frame. The models are learned using standard cross-validation. Pre-images are learned using kernel regressors and have average error $1.7^o$.

**Training Set and Model State Representation:** For training we gather pairs of 3D human poses together with their image projections, here silhouettes, using the graphics package Maya. We use realistically rendered computer graphics human surface models which we animate using human motion capture [1]. Our original human representation ($\mathbf{x}$) is based on articulated skeletons with spherical joints and has 56 skeletal d.o.f. including global translation. The database consists of 8000 samples of human activities including walking, running, turns, jumps, gestures in conversations, quarreling and pantomime.

**Image Descriptors:** We work with image silhouettes obtained using statistical background subtraction (with foreground and background models). Silhouettes are informative for pose estimation although prone to ambiguities (*e.g.* the left / right limb assignment in side views) or occasional lack of observability of some of the d.o.f. (*e.g.* $180^o$ ambiguities in the global azimuthal orientation for frontal views, *e.g.* fig. 1a). These are multiplied by intrinsic forward / backward monocular ambiguities [16]. As observations $\mathbf{r}$, we use shape contexts extracted on the silhouette [4] (5 radial, 12 angular bins, size range 1/8 to 3 on log scale).

The features are computed at different scales and sizes for points sampled on the silhouette. To work in a common coordinate system, we cluster all features in the training set into $K = 50$ clusters. To compute the representation of a new shape feature (a point on the silhouette), we 'project' onto the common basis by (inverse distance) weighted voting into the cluster centers. To obtain the representation ($\mathbf{r}$) for a new silhouette we regularly sample 200 points on it and add all their feature vectors into a feature histogram. Because the representation uses *overlapping features of the observation* the elements of the descriptor are not independent. However, a conditional temporal framework (fig. 1b) flexibly accommodates this.

For experiments, we use Gaussian kernels for the joint angle feature space and dot product kernels for the observation feature space. We learn state conditionals for $p(\mathbf{y}_t|\mathbf{z}_t)$ and $p(\mathbf{y}_t|\mathbf{y}_{t-1}, \mathbf{z}_t)$ using 6 dimensions for the joint angle kernel induced state space and 25 dimensions for the observation induced feature space, respectively. In fig. 3b) we show an evaluation of the efficacy of our kBME predictor for different dimensions in the joint angle kernel induced state space (the observation feature space dimension is here 50). On the analyzed dancing sequence, that involves complex motions of the arms and the legs, the non-linear model significantly outperforms alternative PCA methods and gives good predictions for compact, low-dimensional models.[1]

In tables 1 and 2, as well as fig. 4, we perform quantitative experiments on artificially rendered silhouettes. 3D ground truth joint angles are available and this allows a more

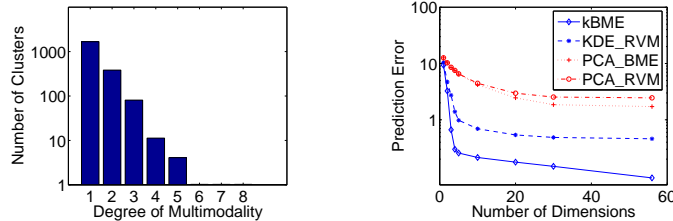

Figure 3: *(a, Left)* Analysis of 'multimodality' for a training set. The input $\mathbf{z}_t$ dimension is 25, the output $\mathbf{y}_t$ dimension is 6, both reduced using kPCA. We cluster independently in $(\mathbf{y}_{t-1}, \mathbf{z}_t)$ and $\mathbf{y}_t$ using many clusters (2100) to simulate small input perturbations and we histogram the $\mathbf{y}_t$ clusters falling within each cluster in $(\mathbf{y}_{t-1}, \mathbf{z}_t)$. This gives intuition on the degree of ambiguity in modeling $p(\mathbf{y}_t|\mathbf{y}_{t-1}, \mathbf{z}_t)$, for small perturbations in the input. *(b, Right)* Evaluation of dimensionality reduction methods for an artificial dancing sequence (models trained on 300 samples). The kBME is our model §2.2, whereas the KDE-RVM is a KDE model learned with a Relevance Vector Machine (RVM) [17] feature space map. PCA-BME and PCA-RVM are models where the mappings between feature spaces (obtained using PCA) is learned using a BME and a RVM. The non-linearity is significant. Kernel-based methods outperform PCA and give low prediction error for 5-6d models.

systematic evaluation. Notice that the kernelized low-dimensional models generally outperform the PCA ones. At the same time, they give results competitive to the ones of high-dimensional BME predictors, while being lower-dimensional and therefore significantly less expensive for inference, *e.g.* the integral in (1).

In fig. 5 and fig. 6 we show human motion reconstruction results for two real image sequences. Fig. 5 shows the good quality reconstruction of a person performing an agile jump. (Given the missing observations in a side view, 3D inference for the occluded body parts would not be possible without using prior knowledge!) For this sequence we do inference using conditionals having 5 modes and reduced 6d states. We initialize tracking using $p(\mathbf{y}_t|\mathbf{z}_t)$, whereas for inference we use $p(\mathbf{y}_t|\mathbf{y}_{t-1}, \mathbf{z}_t)$ within (1). In the second sequence in fig. 6, we simultaneously reconstruct the motion of two people mimicking domestic activities, namely washing a window and picking an object. Here we do inference over a product, 12-dimensional state space consisting of the joint 6d state of each person. We obtain good 3D reconstruction results, using only 5 hypotheses. Notice however, that the results are not perfect, there are small errors in the elbow and the bending of the knee for the subject at the l.h.s., and in the different wrist orientations for the subject at the r.h.s. This reflects the bias of our training set.

| | KDE-RR | RVM | KDE-RVM | BME | kBME |
|---|---|---|---|---|---|
| Walk and turn | 10.46 | 4.95 | 7.57 | 4.27 | 4.69 |
| Conversation | 7.95 | 4.96 | 6.31 | 4.15 | 4.79 |
| Run and turn left | 5.22 | 5.02 | 6.25 | 5.01 | 4.92 |

Table 1: Comparison of average joint angle prediction error for different models. All kPCA-based models use 6 output dimensions. Testing is done on 100 video frames for each sequence, the inputs are artificially generated silhouettes, not in the training set. 3D joint angle ground truth is used for evaluation. KDE-RR is a KDE model with ridge regression (RR) for the feature space mapping, KDE-RVM uses an RVM. BME uses a Bayesian mixture of experts with no dimensionality reduction. kBME is our proposed model. kPCA-based methods use kernel regressors to compute pre-images.

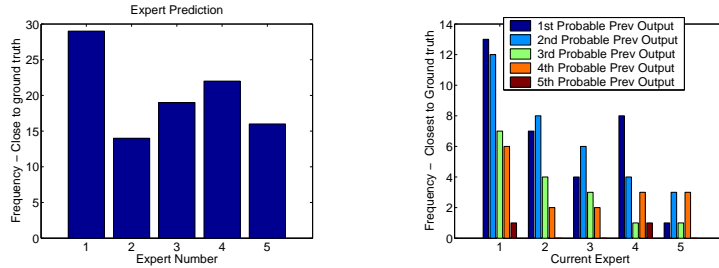

Figure 4: *(a, Left)* Histogram showing the accuracy of various expert predictors: how many times the expert ranked as the $k$-th most probable by the model (horizontal axis) is closest to the ground truth. The model is consistent (the most probable expert indeed is the most accurate most frequently), but occasionally less probable experts are better. *(b, Right)* Histograms show the dynamics of $p(\mathbf{y}_t|\mathbf{y}_{t-1}, \mathbf{z}_t)$, *i.e.* how the probability mass is redistributed among experts between two successive time steps, in a conversation sequence.

|  | KDE-RR | RVM | KDE-RVM | BME | kBME |
|---|---|---|---|---|---|
| Walk and turn back | 7.59 | 6.9 | 7.15 | 3.6 | 3.72 |
| Run and turn | 17.7 | 16.8 | 16.08 | 8.2 | 8.01 |

Table 2: Joint angle prediction error computed for two complex sequences with walks, runs and turns, thus more ambiguity (100 frames). Models have 6 state dimensions. Unimodal predictors average competing solutions. kBME has significantly lower error.

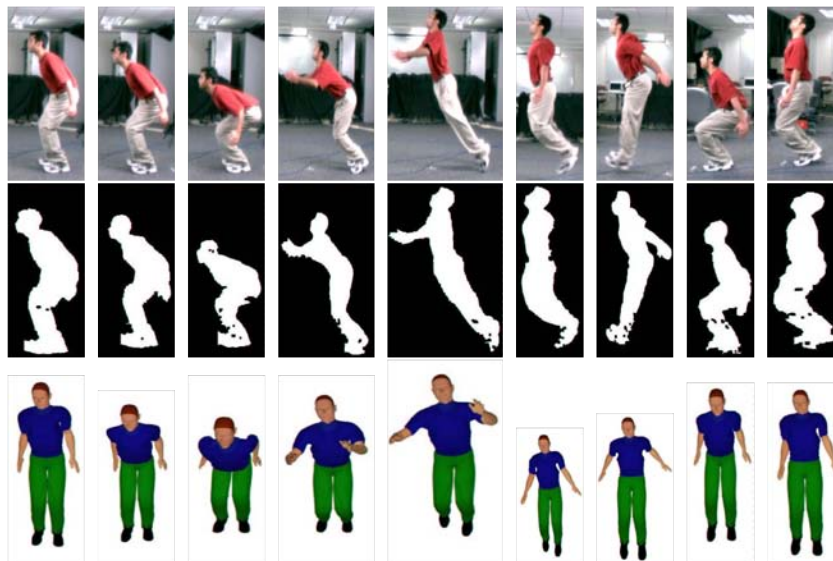

Figure 5: Reconstruction of a jump (selected frames). *Top:* original image sequence. *Middle:* extracted silhouettes. *Bottom:* 3D reconstruction seen from a synthetic viewpoint.

## 4 Conclusion

We have presented a probabilistic framework for conditional inference in latent kernel-induced low-dimensional state spaces. Our approach has the following properties: *(a)*

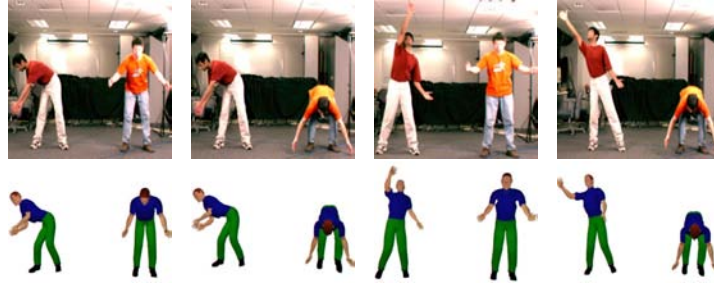

Figure 6: Reconstructing the activities of 2 people operating in an 12-d state space (each person has its own 6d state). *Top:* original image sequence. *Bottom:* 3D reconstruction seen from a synthetic viewpoint.

Accounts for non-linear correlations among input or output variables, by using kernel non-linear dimensionality reduction (kPCA); *(b)* Learns probability distributions over mappings between low-dimensional state spaces using conditional Bayesian mixture of experts, as required for accurate prediction. In the resulting low-dimensional kBME predictor ambiguities and multiple solutions common in visual, inverse perception problems are accurately represented. *(c)* Works in a continuous, conditional temporal probabilistic setting and offers a formal management of uncertainty. We show comparisons that demonstrate how the proposed approach outperforms regression, PCA or KDE alone for reconstructing the 3D human motion in monocular video. Future work we will investigate scaling aspects for large training sets and alternative structured prediction methods.

## Footnotes

[1]**Running times:** On a Pentium 4 PC (3 GHz, 2 GB RAM), a full dimensional BME model with 5 experts takes 802s to train $p(\mathbf{x}_t|\mathbf{x}_{t-1}, \mathbf{r}_t)$, whereas a kBME (including the pre-image) takes 95s to train $p(\mathbf{y}_t|\mathbf{y}_{t-1}, \mathbf{z}_t)$. The prediction time is 13.7s for BME and 8.7s (including the pre-image cost 1.04s) for kBME. The integration in (1) takes 2.67s for BME and 0.31s for kBME. The speed-up for kBME is significant and likely to increase with original models having higher dimensionality.

## References

[1] CMU Human Motion DataBase. Online at http://mocap.cs.cmu.edu/search.html, 2003.

[2] A. Agarwal and B. Triggs. 3d human pose from silhouettes by Relevance Vector Regression. In *CVPR*, 2004.

[3] G. Bakir, J. Weston, and B. Scholkopf. Learning to find pre-images. In *NIPS*, 2004.

[4] S. Belongie, J. Malik, and J. Puzicha. Shape matching and object recognition using shape contexts. *PAMI*, 24, 2002.

[5] C. Bishop and M. Svensen. Bayesian mixtures of experts. In *UAI*, 2003.

[6] J. Deutscher, A. Blake, and I. Reid. Articulated Body Motion Capture by Annealed Particle Filtering. In *CVPR*, 2000.

[7] M. Jordan and R. Jacobs. Hierarchical mixtures of experts and the EM algorithm. *Neural Computation*, (6):181–214, 1994.

[8] D. Mackay. Bayesian interpolation. *Neural Computation*, 4(5):720–736, 1992.

[9] A. McCallum, D. Freitag, and F. Pereira. Maximum entropy Markov models for information extraction and segmentation. In *ICML*, 2000.

[10] R. Rosales and S. Sclaroff. Learning Body Pose Via Specialized Maps. In *NIPS*, 2002.

[11] B. Schölkopf, A. Smola, and K. Müller. Nonlinear component analysis as a kernel eigenvalue problem. *Neural Computation*, 10:1299–1319, 1998.

[12] G. Shakhnarovich, P. Viola, and T. Darrell. Fast Pose Estimation with Parameter Sensitive Hashing. In *ICCV*, 2003.

[13] L. Sigal, S. Bhatia, S. Roth, M. Black, and M. Isard. Tracking Loose-limbed People. In *CVPR*, 2004.

[14] C. Sminchisescu and A. Jepson. Generative Modeling for Continuous Non-Linearly Embedded Visual Inference. In *ICML*, pages 759–766, Banff, 2004.

[15] C. Sminchisescu, A. Kanaujia, Z. Li, and D. Metaxas. Discriminative Density Propagation for 3D Human Motion Estimation. In *CVPR*, 2005.

[16] C. Sminchisescu and B. Triggs. Kinematic Jump Processes for Monocular 3D Human Tracking. In *CVPR*, volume 1, pages 69–76, Madison, 2003.

[17] M. Tipping. Sparse Bayesian learning and the Relevance Vector Machine. *JMLR*, 2001.

[18] C. Tomasi, S. Petrov, and A. Sastry. 3d tracking = classification + interpolation. In *ICCV*, 2003.

[19] J. Weston, O. Chapelle, A. Elisseeff, B. Scholkopf, and V. Vapnik. Kernel dependency estimation. In *NIPS*, 2002.
